# Graded grammaticality in Prediction Fractal Machines

**Shan Parfitt, Peter Tiňo and Georg Dorffner**
Austrian Research Institute for Artificial Intelligence,
Schottengasse 3, A-1010 Vienna, Austria.
{*shan,petert,georg*}*@ai.univie.ac.at*

## Abstract

We introduce a novel method of constructing language models, which avoids some of the problems associated with recurrent neural networks. The method of creating a Prediction Fractal Machine (PFM) [1] is briefly described and some experiments are presented which demonstrate the suitability of PFMs for language modeling. PFMs distinguish reliably between minimal pairs, and their behavior is consistent with the hypothesis [4] that wellformedness is 'graded' not absolute. A discussion of their potential to offer fresh insights into language acquisition and processing follows.

## 1 Introduction

Cognitive linguistics has seen the development in recent years of two important, related trends. Firstly, a widespread renewal of interest in the statistical, 'graded' nature of language (e.g. [2]-[4]) is showing that the traditional all-or-nothing notion of well-formedness may not present an accurate picture of how the congruity of utterances is represented internally. Secondly, the analysis of state space trajectories in artificial neural networks (ANNs) has provided new insights into the types of processes which may account for the ability of learning devices to acquire and represent language, without appealing to traditional linguistic concepts [5]-[7]. Despite the remarkable advances which have come out of connectionist research (e.g. [8]), and the now common use of recurrent networks, and Simple Recurrent Networks (SRNs) [9] especially, in the study of language (e.g. [10]), recurrent neural networks suffer from particular problems which make them imperfectly suited to language tasks. The vast majority of work in this field employs small networks and datasets (usually artificial), and although many interesting linguistic issues may be thus tackled, real progress in evaluating the potentials of state trajectories and graded 'grammaticality' to uncover the underlying processes responsible for overt linguistic phenomena must inevitably be limited whilst the experimental tasks remain so small. Nevertheless, there are certain obstacles to the scaling-up of networks trained by back-propagation (BP). Such networks tend towards ever

longer training times as the sizes of the input set and of the network increase, and although Real-Time Recurrent Learning (RTRL) and Back-propagation Through Time are potentially better at modeling temporal dependencies, training times are longer still [11]. Scaling-up is also difficult due to the potential for catastrophic interference and lack of adaptivity and stability [12]-[14]. Other problems include the rapid loss of information about past events as the distance from the present increases [15] and the dependence of learned state trajectories not only on the training data, but also upon such vagaries as initial weight vectors, making their analysis difficult [16]. Other types of learning device also suffer problems. Standard Markov models require the allocation of memory for every $n$-gram, such that large values of $n$ are impractical; variable-length Markov models are more memory-efficient, but become unmanageable when trained on large data sets [17]. Two important, related concerns in cognitive linguistics are thus (a) to find a method which allows language models to be scaled up, which is similar in spirit to recurrent neural networks, but which does not encounter the same problems of scale, and (b) to use such a method to evince new insights into graded grammaticality from the state trajectories which arise given genuinely large, naturally-occurring data sets.

Accordingly, we present a new method of generating state trajectories which avoids most of these problems. Previously studied in a financial prediction task, the method creates a fractal map of the training data, from which state machines are built. The resulting models are known as Prediction Fractal Machines (PFMs) [18] and have some useful properties. The state trajectories in the fractal representation are fast and computationally efficient to generate, and are accurate and well-understood; it may be inferred that, even for very large vocabularies and training sets, catastrophic interference and lack of adaptivity and stability will not be a problem, given the way in which representations are built (demonstrating this is a topic for future work); training times are significantly less than for recurrent networks (in the experiments described below, the smallest models took a few minutes to build, while the largest ones took only around three hours; in comparison, all of the ANNs took longer - up to a day - to train); and there is little or no loss of information over the course of an input sequence (allowing for the finite precision of the computer). The scalability of the PFM was taken advantage of by training on a large corpus of naturally-occurring text. This enabled an assessment of what potential new insights might arise from the use of this method in truly large-scale language tasks.

## 2 Prediction Fractal Machines (PFMs)

A brief description of the method of creating a PFM will now be given. Interested readers should consult [1], since space constraints preclude a detailed examination here. The key idea behind our predictive model is a transformation $F$ of symbol sequences from an alphabet (here, tagset) $\{1, 2, ..., N\}$ into points in a hypercube $H = [0, 1]^D$. The dimensionality $D$ of the hypercube $H$ should be large enough for each symbol $1, 2, ..., N$ to be identified with a *unique* vertex of $H$. The particular assignment of symbols to vertices is arbitrary. The transformation $F$ has the crucial property that symbol sequences sharing the same suffix (context) are mapped close to each other. Specifically, the longer the common suffix shared by two sequences, the smaller the (Euclidean) distance between their point representations. The transformation $F$ used in this study corresponds to an Iterative Function System [19]

consisting of $N$ affine maps $i : H \to H$, $i = 1, 2, ..., N$,

$$i(x) = \frac{1}{2}(x + t_i), \quad t_i \in \{0, 1\}^D, \quad t_i \neq t_j \text{ for } i \neq j. \tag{1}$$

Given a sequence $s_1 s_2 ... s_L$ of $L$ symbols from the alphabet $1, 2, ..., N$, we construct its point representation as

$$s_L(s_{L-1}(...(s_2(s_1(x^*)))...)) = (s_L \circ s_{L-1} \circ ... \circ s_2 \circ s_1)(x^*), \tag{2}$$

where $x^*$ is the center $\{\frac{1}{2}\}^D$ of the hypercube $H$. (Note that as is common in the Iterative Function Systems literature, $i$ refers either to a symbol or to a map, depending upon the context.) PFMs are constructed on point representations of subsequences appearing in the training sequence. First, we slide the window of length $L > 1$ over the training sequence. At each position we transform the sequence of length $L$ appearing in the window into a point. The set of points obtained by sliding through the whole training sequence is then partitioned into several classes by k-means vector quantization (in the Euclidean space), each class represented by a particular codebook vector. The number of codebook vectors required is chosen experimentally. Since quantization classes group points lying close together, sequences having point representations in the same class (potentially) share long suffixes. The quantization classes may then be treated as prediction contexts, and the corresponding predictive symbol probabilities computed by sliding the window over the training sequence again and counting, for each quantization class, how often a sequence mapped to that class was followed by a particular symbol. In test mode, upon seeing a new sequence of $L$ symbols, the transformation $F$ is again performed, the closest quantization center found, and the corresponding predictive probabilities used to predict the next symbol.

## 3   An experimental comparison of PFMs and recurrent networks

The performance of the PFM was compared against that of a RTRL-trained recurrent network on a next-tag prediction task. Sixteen grammatical tags and a 'sentence start' character were used. The models were trained on a concatenated sequence (22781 tags) of the top three-quarters of each of the 14 sub-corpora of the University of Pennsylvania 'Brown' corpus[1]. The remainder was used to create test data, as follows. Because in a large training corpus of naturally-occurring data, contexts in most cases have more than one possible correct continuation, simply counting correctly predicted symbols is insufficient to assess performance, since this fails to count correct responses which are not targets. The extent to which the models distinguished between grammatical and ungrammatical utterances was therefore additionally measured by generating minimal pairs and comparing their negative log likelihoods (NLLs) per symbol with respect to the model. Likelihood is computed by sliding through the test sequence and for each window position, determining the probability of the symbol that appears immediately beyond it. As processing progresses, these probabilities are multiplied. The negative of the natural logarithm is then taken and divided by the number of symbols. Significant differences in NLLs

are much harder to achieve between members of minimal pairs than between grammatical and random sequences, and are therefore a good measure of model validity. Minimal pairs generated by theoretically-motivated manipulations tend to be no longer ungrammatical given a small tagset, because the removal of grammatical sub-classes necessarily also removes a large amount of information. Manipulations were therefore performed by switching the positions of two symbols in each sentence in the test sets. Symbols switched could be any distance apart within the sentence, as long as the resulting sentence was ungrammatical under all surface instantiations. By changing as little as possible to make the sentence ungrammatical, the goal was retained that the task of distinguishing between grammatical and ungrammatical sequences be as difficult as possible. The test data then consisted of 28 paired grammatical/ungrammatical test sets (around 570 tags each), plus an ungrammatical, 'meaningless' test set containing all 17 codes listed several times over, used to measure baseline performance. Ten 1st-order randomly-initialised networks were trained for 100 epochs using RTRL. The networks consisted of 1 input and 1 output layer, each with 17 units corresponding to the 17 tags, 2 hidden layers, each with 10 units, and 1 context layer of 10 units connected to the first hidden layer. The second hidden layer was used to increase the flexibility of the maps between the hidden representations in the recurrent portion and the tag activations at the output layer. A logistic sigmoid activation function was used, the learning rate and momentum were set to 0.05, and the training sequence was presented at the rate of one tag per clock tick. The PFMs were derived by clustering the fractal representation of the training data ten times for various numbers of codebook vectors between 5 and 200. More experiments were performed using PFMs than neural networks because in the former case, experience in choosing appropriate numbers of codebook vectors was initially lacking for this type of data.

The results which follow are given as averages, either over all neural networks, or else over all PFMs derived from a given number of codebook vectors. The networks correctly predicted 36.789% and 32.667% of next tags in the grammatical and ungrammatical test sets, respectively. The PFMs matched this performance at around 30 codebook vectors (37.134% and 32.814% respectively), and exceeded it for higher numbers of vectors (39.515% and 34.388% respectively at 200 vectors). The networks generated mean NLLs per symbol of 1.966 and 2.182 for the grammatical and ungrammatical test sets, respectively (a difference of 0.216) and 4.157 for the 'meaningless' test set (the difference between NLLs for grammatical and 'meaningless' data = 2.191). The PFMs matched this difference in NLLs at 40 codebook vectors (NLL grammatical = 1.999, NLL ungrammatical = 2.217; difference = 0.218). The NLL for the 'meaningless' data at 40 codebook vectors was 6.075 (difference between NLLs for grammatical and 'meaningless' data = 4.076). The difference between NLLs for grammatical and ungrammatical, and for grammatical and 'meaningless' data sets, became even larger with increased numbers of codebook vectors. The difference in performance between grammatical and ungrammatical test sets was thus highly significant in all cases ($p < .0005$): all the models distinguished what was grammatical from what was not. This conclusion is supported by the fact that the mean NLLs for the 'meaningless' test set were always noticeably higher than those for the minimal pair sets.

## 4   Discussion

The PFMs exceeded the performance of the networks for larger numbers of code-book vectors, but it is possible that networks with more hidden nodes would also do better. In terms of ease of use, however, as well as in their scaling-up potential, PFMs are certainly superior. Their other great advantage is that the representations created are dependable (see section 1), making hypothesis creation and testing not just more rapid, but also more straightforward: the speed with which PFMs may be trained made it possible to make statistically significant observations for a large number of clustering runs. In the introduction, 'graded' wellformedness was spoken of as being productive of new hypotheses about the nature of language. Our use of minimal pairs, designed to make a clear-cut distinction between grammatical and ungrammatical utterances, appears to leave this issue to one side. But in reality, our results were rather pertinent to it, as the use of the likelihood measure might indeed imply. The Brown corpus consists of subcorpora representative of 14 different discourse types, from fiction to government documents. Whereas traditional notions of grammaticality would lead us to treat all of the 'ungrammatical' sentences in the minimal pair test sets as equally ungrammatical, the NLLs in our experiments tell a different story. The grammatical versions consistently had a lower associated NLL (higher probability) than the ungrammatical versions, but the difference between these was much smaller than that between the 'meaningless' data and either the grammatical or the ungrammatical data. This supports the concept of 'graded grammaticality', and NLLs for 'meaningless' data such as ours might be seen as a sort of benchmark by which to measure all lesser degrees of ungrammaticality. (Note incidentally that the PFMs appear to associate with the 'meaningless' data a significantly higher NLL than did the networks, even though the difference between the NLLs of the grammatical and ungrammatical data was the same. This is suggestive of PFMs having greater powers of discrimination between grades of wellformedness than the recurrent networks used, but further research will be needed to ascertain the validity of this.) Moreover, the NLL varied not just between grammatical and ungrammatical test sets, but also from sentence to sentence, from word to word and from discourse style to discourse style. While it increased, often dramatically, when the manipulated portion of an ungrammatical sentence was encountered, some words in grammatical sentences exhibited a similar effect: thus, if a subsequence in a well-formed utterance occurs only rarely - or never - in a training set, it will have a high associated NLL in the same way as an ungrammatical one does. This is likely to happen even for very large corpora, since some grammatical structures are very rare. This is consistent with recent findings that, during human sentence processing, well-formedness is linked to conformity with expectation [20] as measured by CLOZE scores. Interesting also was the remarkable variation in NLL between discourse styles. Although the mean NLL across all discourse styles (test sets) is lower for the grammatical than for the ungrammatical versions, it cannot be guaranteed that the grammatical version of one test set will have a lower NLL than the ungrammatical version of another. Indeed, the grammatical and ungrammatical NLLs interleave, as may be observed in figure 1, which shows the NLLs for the three discourse styles which lie at the bottom, middle and top of the range. Even more interestingly, if the NLLs for the grammatical versions of all discourse styles are ordered according to where they lie within this range, it becomes clear that NLL is a predictor of discourse style. Styles which linguists class as 'formal', e.g. those of

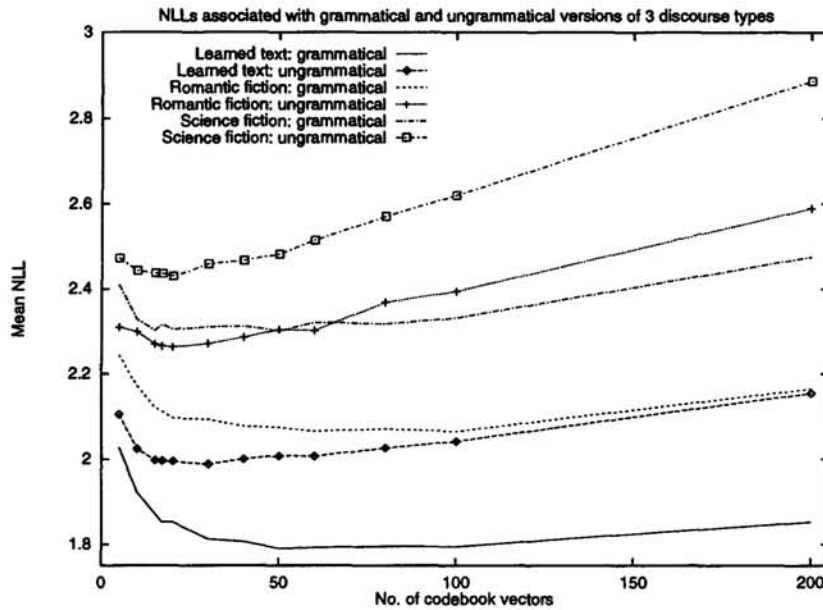

Figure 1: NLLs of minimal pair test sets containing different discourse styles suggest grades of wellformedness based upon prototypicality.

the Learned and Government Document test sets, have the lowest NLLs, with the three Press test sets clustering just above, and the Fiction test sets, exemplifying creative language use, clustering at the high end. Similarly, that the Learned and Government test sets have the lowest NLLs conforms with the intuition that their usage lies closest to what is grammatically 'prototypical' - even though in the training set, 6 out of the 14 test sets are fiction and thus might be expected to contribute more to the prototype. That they do not, suggests that usage varies significantly across fiction test sets.

## 5 Conclusion

Work on the use of PFMs in language modeling is at an early stage, but as results to date show, they have a lot to offer. A much larger project is planned, which will examine further Allen and Seidenberg's hypothesis that 'graded grammaticality' (or wellformedness) applies not only to syntax, but also to other language subdomains such as semantics, an integral part of this being the use of larger corpora and tagsets, and the identification of vertices with semantic/syntactic features rather than atomic symbols. Identifying the possibilities of combining PFMs with ANNs, for example as a means of bypassing the normal method of creating state-space trajectories, is the subject of current study.

### Acknowledgments

This work was supported by the Austrian Science Fund (FWF) within the research project "Adaptive Information Systems and Modeling in Economics and Management Science" (SFB 010). The Austrian Research Institute for Artificial Intelligence is supported by the Austrian Federal Ministry of Science and Transport.

## Footnotes

[1] http://www.ldc.upenn.edu/

# References

[1] P. Tiňo & G. Dorffner (1998). Constructing finite-context sources from fractal representations of symbolic sequences. Technical Report TR-98-18, Austrian Research Institute for AI, Vienna.

[2] J. R. Taylor (1995). *Linguistic categorisation: Prototypes in linguistic theory.* Clarendon, Oxford.

[3] J. R. Saffran, R. N. Aslin & E. L. Newport (1996). Statistical cues in language acquisition: Word segmentation by infants. In *Proc. of the Cognitive Science Society Conference*, 376–380, La Jolla, CA.

[4] J. Allen & M. S. Seidenberg (in press). The emergence of grammaticality in connectionist networks. In B. Macwhinney (ed.), *Emergentist approaches to language: Proc. of the 28th Carnegie Symposium on cognition.* Erlbaum.

[5] S. Parfitt (1997). *Aspects of anaphora resolution in artificial neural networks: Implications for nativism.* PhD thesis, Imperial College, London.

[6] D. Servan-Schreiber *et al* (1989). Graded state machines: The representation of temporal contingencies in Simple Recurrent Networks. In *Advances in Neural Information Processing Systems*, 643–652.

[7] W. Tabor & M. Tanenhaus (to appear). Dynamical models of sentence processing. Cognitive Science.

[8] J. L. Elman *et al* (1996). *Rethinking innateness: A connectionist perspective on development.* Bradford.

[9] J. L. Elman (1990). Finding structure in time. In: Cognitive Science, 14: 179-211.

[10] S. Lawrence, C. Lee Giles & S. Fong (in press). Natural language grammatical inference with recurrent neural networks. *IEEE Trans. on knowledge and data engineering.*

[11] J. Hertz, A. Krogh & R. G. Palmer (1991). *Introduction to the theory of neural computation.* Addison Wesley.

[12] M. McCloskey & N. J. Cohen (1989). Catastrophic interference in connectionist networks: The sequential learning problem. In G. Bower (ed.), *The psychology of learning and motivation, vol 24.* Academic, NY.

[13] J. K. Kruschke (1991). ALCOVE: A connectionist model of human category learning. In R. P. Lippman *et al* (eds.), *Advances in Neural Information Processing 3*, 649–655. Kaufmann, San Mateo, CA.

[14] S. Grossberg (ed.) (1988). *Neural networks and natural intelligence.* Bradford, MIT, Cambs, MA.

[15] Y. Bengio, P. Simard & P. Frasconi (1994). Learning long-term dependencies with gradient descent is difficult. *IEEE Trans. on neural networks,* 5(2).

[16] M. P. Casey (1996). The dynamics of discrete-time computation, with application to recurrent neural networks and finite-state machine extraction. *Neural Computation,* 8(6):1135–1178.

[17] D. Ron, Y. Singer & N. Tishby (1996). The power of amnesia. *Machine Learning,* 25.

[18] P. Tiňo, B. G. Horne, C. Lee Giles & P. C. Collingwood (1998). Finite state machines and recurrent neural networks - automata and dynamical systems approaches. In J. E. Dayhoff & O. Omidvar (eds.), *Neural Networks and Pattern Recognition*, 171–220. Academic.

[19] M. F. Barnsley (1988). *Fractals everywhere.* Academic, NY.

[20] S. Coulson, J. W. King & M. Kutas (1998). Expect the unexpected: Responses to morphosyntactic violations. *Language and Cognitive Processes*, 13(1).
